# Maximising Sensitivity in a Spiking Network

**Anthony J. Bell,**
Redwood Neuroscience Institute
1010 El Camino Real, Suite 380
Menlo Park, CA 94025
tbell@rni.org

**Lucas C. Parra**
Biomedical Engineering Department
City College of New York
New York, NY 10033
parra@ccny.cuny.edu

## Abstract

We use unsupervised probabilistic machine learning ideas to try to explain the kinds of learning observed in real neurons, the goal being to connect abstract principles of self-organisation to known biophysical processes. For example, we would like to explain Spike Timing-Dependent Plasticity (see [5,6] and Figure 3A), in terms of information theory. Starting out, we explore the optimisation of a network *sensitivity* measure related to maximising the mutual information between input spike timings and output spike timings. Our derivations are analogous to those in ICA, except that the sensitivity of output timings to input timings is maximised, rather than the sensitivity of output 'firing rates' to inputs. ICA and related approaches have been successful in explaining the learning of many properties of early visual receptive fields in rate coding models, and we are hoping for similar gains in understanding of spike coding in networks, and how this is supported, in principled probabilistic ways, by cellular biophysical processes. For now, in our initial simulations, we show that our derived rule can learn synaptic weights which can unmix, or *demultiplex*, mixed spike trains. That is, it can recover independent point processes embedded in distributed correlated input spike trains, using an adaptive single-layer feedforward spiking network.

## 1 Maximising Sensitivity.

In this section, we will follow the structure of the ICA derivation [4] in developing the spiking theory. We cannot claim, as before, that this gives us an information maximisation algorithm, for reasons that we will delay addressing until Section 3. But for now, to first develop our approach, we will explore an interim objective function called *sensitivity* which we define as the log Jacobian of how input spike timings affect output spike timings.

### 1.1 How to maximise the effect of one spike timing on another.

Consider a spike in neuron $j$ at time $t_l$ that has an effect on the timing of another spike in neuron $i$ at time $t_k$. The neurons are connected by a weight $w_{ij}$. We use $i$ and $j$ to index neurons, and $k$ and $l$ to index spikes, but sometimes for convenience we will use spike indices in place of neuron indices. For example, $w_{kl}$, the weight between an input spike $l$ and an output spike $k$, is naturally understood to be just the corresponding $w_{ij}$.

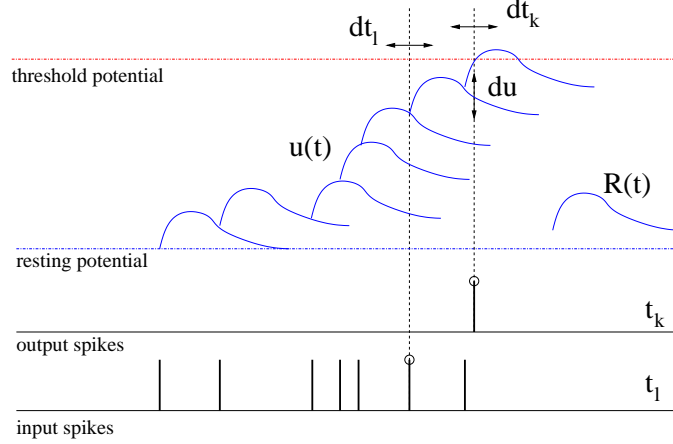

Figure 1: Firing time $t_k$ is determined by the time of threshold crossing. A change of an input spike time $dt_l$ affects, via a change of the membrane potential $du$ the time of the output spike by $dt_k$.

In the simplest version of the Spike Response Model [7], spike $l$ has an effect on spike $k$ that depends on the time-course of the evoked EPSP or IPSP, which we write as $R_{kl}(t_k - t_l)$. In general, this $R_{kl}$ models both synaptic and dendritic linear responses to an input spike, and thus models synapse type and location. For learning, we need only consider the value of this function when an output spike, $k$, occurs.

In this model, depicted in Figure 1, a neuron adds up its spiking inputs until its membrane potential, $u_i(t)$, reaches threshold at time $t_k$. This threshold we will often, again for convenience, write as $u_k \equiv u_i(t_k, \{t_l\})$, and it is given by a sum over spikes $l$:

$$u_k = \sum_l w_{kl} R_{kl}(t_k - t_l) . \tag{1}$$

To maximise timing sensitivity, we need to determine the effect of a small change in the input firing time $t_l$ on the output firing time $t_k$. (A related problem is tackled in [2].) When $t_l$ is changed by a small amount $dt_l$ the membrane potential will change as a result. This change in the membrane potential leads to a change in the time of threshold crossing $dt_k$. The contribution to the membrane potential, $du$, due to $dt_l$ is $(\partial u_k/\partial t_l)dt_l$, and the change in $du$ corresponding to a change $dt_k$ is $(\partial u_k/\partial t_k)dt_k$. We can relate these two effects by noting that the total change of the membrane potential $du$ has to vanish because $u_k$ is defined as the potential at threshold. ie:

$$du = \frac{\partial u_k}{\partial t_k}dt_k + \frac{\partial u_k}{\partial t_l}dt_l = 0 . \tag{2}$$

This is the *total differential* of the function $u_k = u(t_k, \{t_l\})$, and is a special case of the implicit function theorem. Rearranging this:

$$\frac{dt_k}{dt_l} = -\frac{\partial u_k}{\partial t_l} \bigg/ \frac{\partial u_k}{\partial t_k} = -w_{kl}\dot{R}_{kl}/\dot{u}_k . \tag{3}$$

Now, to connect with the standard ICA derivation [4], recall the 'rate' (or sigmoidal) neuron, for which $y_i = g_i(u_i)$ and $u_i = \sum_j w_{ij}x_j$. For this neuron, the output dependence on

input is $\partial y_i / \partial x_j = w_{ij} g_i'$ while the learning gradient is:

$$\frac{\partial}{\partial w_{ij}} \log \left| \frac{\partial y_i}{\partial x_j} \right| = \frac{1}{w_{ij}} - f_i(u_i) x_j \tag{4}$$

where the 'score functions', $f_i$, are defined in terms of a density estimate on the summed inputs: $f_i(u_i) = \frac{\partial}{\partial u_i} \log g_i' = \frac{\partial}{\partial u_i} \log \hat{p}(u_i)$.

The analogous learning gradient for the spiking case, from (3), is:

$$\frac{\partial}{\partial w_{ij}} \log \left| \frac{dt_k}{dt_l} \right| = \frac{1}{w_{ij}} - \frac{\sum_a j(a) \dot{R}_{ka}}{\dot{u}_k} \, . \tag{5}$$

where $j(a) = 1$ if spike $a$ came from neuron $j$, and 0 otherwise.

Comparing the two cases in (4) and (5), we see that the input variable $x_j$ has become the temporal derivative of the sum of the EPSPs coming from synapse $j$, and the output variable (or score function) $f_i(u_i)$ has become $\dot{u}_k^{-1}$, the inverse of the temporal derivative of the membrane potential at threshold. It is intriguing (A) to see this quantity appear as analogous to the score function in the ICA likelihood model, and, (B) to speculate that experiments could show that this' voltage slope at threshold' is a hidden factor in STDP data, explaining some of the scatter in Figure 3A. In other words, an STDP datapoint should lie on a 2-surface in a 3D space of $\{\Delta w, \ \Delta t, \ \dot{u}_k\}$. Incidentally, $\dot{u}_k$ shows up in any learning rule optimising an objective function involving output spike timings.

## 1.2 How to maximise the effect of $N$ spike timings on $N$ other ones.

Now we deal with the case of a 'square' single-layer feedforward mapping between spike timings. There can be several input and output neurons, but here we ignore which neurons are spiking, and just look at how the input timings affect the output timings. This is captured in a Jacobian matrix of all timing dependencies we call $\mathbf{T}$. The entries of this matrix are $\mathbf{T}_{kl} \equiv \partial t_k / \partial t_l$. A multivariate version of the *sensitivity* measure introduced in the previous section is the log of the absolute determinant of the timing matrix, ie: $\log |\mathbf{T}|$. The full derivation for the gradient $\nabla_{\mathbf{W}} \log |\mathbf{T}|$ is in the Appendix. Here, we again draw out the analogy between Square ICA [4] and this gradient, as follows. Square ICA with a network $\mathbf{y} = \mathbf{g}(\mathbf{W}\mathbf{x})$ is:

$$\Delta \mathbf{W} \propto \nabla_{\mathbf{W}} \log |\mathbf{J}| = \mathbf{W}^{-1} - \mathbf{f}(\mathbf{u}) \mathbf{x}^T \tag{6}$$

where the Jacobian $\mathbf{J}$ has entries $\partial y_i / \partial x_j$ and the score functions are now, $f_i(\mathbf{u}) = -\frac{\partial}{\partial u_i} \log \hat{p}(\mathbf{u})$ for the general likelihood case, with $\hat{p}(\mathbf{u}) = \prod_i g_i'$ being the special case of ICA. We will now split the gradient in (6) according to the chain rule:

$$\nabla_{\mathbf{W}} \log |\mathbf{J}| = [\nabla_{\mathbf{J}} \log |\mathbf{J}|] \otimes [\nabla_{\mathbf{W}} \mathbf{J}] \tag{7}$$

$$= \left[ \mathbf{J}^{-T} \right] \otimes \left[ J_{kl} \, i(k) \left( \frac{j(l)}{w_{kl}} - f_k(\mathbf{u}) x_j \right) \right] \, . \tag{8}$$

In this equation, $i(k) = \delta_{ik}$ and $j(l) = \delta_{jl}$. The righthand term is a 4-tensor with entries $\partial J_{kl} / \partial w_{ij}$, and $\otimes$ is defined as $\mathbf{A} \otimes \mathbf{B}_{ij} = \sum_{kl} A_{kl} B_{klij}$. We write the gradient this way to preserve, in the second term, the independent structure of the $1 \rightarrow 1$ gradient term in (4), and to separate a difficult derivation into two easy parts. The structure of (8) holds up when we move to the spiking case, giving:

$$\nabla_{\mathbf{W}} \log |\mathbf{T}| = [\nabla_{\mathbf{T}} \log |\mathbf{T}|] \otimes [\nabla_{\mathbf{W}} \mathbf{T}] \tag{9}$$

$$= \left[ \mathbf{T}^{-T} \right] \otimes \left[ T_{kl} \, i(k) \left( \frac{j(l)}{w_{kl}} - \frac{\sum_a j(a) \dot{R}_{ka}}{\dot{u}_k} \right) \right] \tag{10}$$

where $i(k)$ is now defined as being 1 if spike $k$ occured in neuron $i$, and 0 otherwise. $j(l)$ and $j(a)$ are analogously defined.

Because the $\mathbf{T}$ matrix is much bigger than the $\mathbf{J}$ matrix, and because it's entries are more complex, here the similarity ends. When (10) is evaluated for a single weight influencing a single spike coupling (see the Appendix for the full derivation), it yields:

$$\Delta w_{kl} \propto \frac{\partial \log |\mathbf{T}|}{\partial w_{kl}} = \frac{T_{kl}}{w_{kl}} \left( \left[ \mathbf{T}^{-1} \right]_{lk} - 1 \right) , \tag{11}$$

This is a non-local update involving a matrix inverse at each step. In the ICA case of (6), such an inverse was removed by the Natural Gradient transform (see [1]), but in the spike timing case, this has turned out not to be possible, because of the additional asymmetry introduced into the $\mathbf{T}$ matrix (as opposed to the $\mathbf{J}$ matrix) by the $\dot{R}_{kl}$ term in (3).

## 2 Results.

Nonetheless, this learning rule can be simulated. It requires running the network for a while to generate spikes (and a corresponding $\mathbf{T}$ matrix), and then for each input/output spike coupling, the corresponding synapse is updated according to (11). When this is done, and the weights learn, it is clear that something has been sacrificed by ignoring the issue of which neurons are producing the spikes. Specifically, the network will often put all the output spikes on one output neuron, with the rates of the others falling to zero. It is happy to do this, if a large $\log |\mathbf{T}|$ can thereby be achieved, because we have not included this 'which neuron' information in the objective. We will address these and other problems in Section 3, but now we report on our simulation results on demultiplexing.

### 2.1 Demultiplexing spike trains.

An interesting possibility in the brain is that 'patterns' are embedded in spatially distributed spike timings that are input to neurons. Several patterns could be embedded in single input trains. This is called *multiplexing*. To extract and propagate these patterns, the neurons must *demultiplex* these inputs using its threshold nonlinearity. Demultiplexing is the 'point process' analog of the unmixing of independent inputs in ICA. We have been able to robustly achieve demultiplexing, as we now report.

We simulated a feed-forward network with 3 integrate-and-fire neurons and inputs from 3 presynaptic neurons. Learning followed (11) where we replace the inverse by the pseudo-inverse computed on the spikes generated during 0.5 s. The pseudo-inverse is necessary because even though on average, the learning matches number of output spikes to number of input spikes, the matrix $\mathbf{T}$ is still not usually square and so its actual inverse cannot be taken.

In addition, in these simulations, an additional term is introduced in the learning to make sure all the output neurons fire with equal probability. This partially counters the ignoral of the 'which neuron' information, which we explained above. Assuming Poisson spike count $n_i$ for the $i$th output neuron with equal firing rate $\bar{n}_i$ it is easy to derive in an approximate term that will control the spike count, $\sum_i (\bar{n}_i - n_i)$. The target firing rates $\bar{n}_i$ were set to match the "source" spike train in this example.

The network learns to demultiplex mixed spike trains, as shown in Figure 2. This demultiplexing is a robust property of learning using (11) with this new spike-controlling term.

Finally, what about the spike-timing dependence of the observed learning? Does it match experimental results? The comparison is made in Figure 3, and the answer is no. There is a timing-dependent transition between depression and potentiation in our result

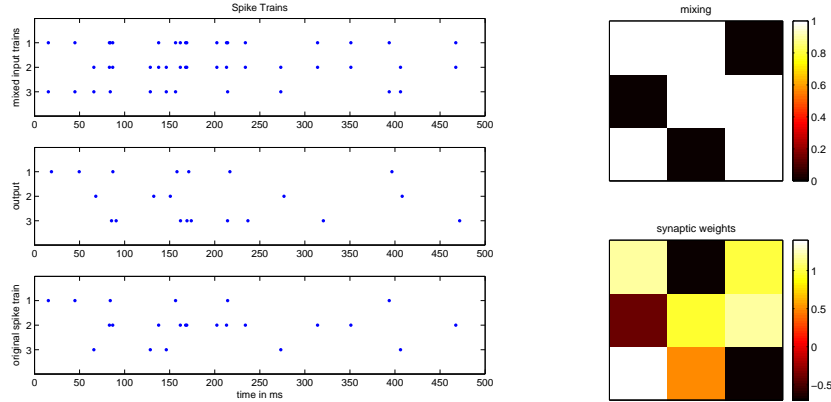

Figure 2: Unmixed spike trains. The input (top lef) are 3 spike trains which are a mixture of three independent Poison processes (bottom left). The network unmixes the spike train to approximately recover the original (center left). In this example 19 spikes correspond to the original with 4 deletion and 2 insertions. The two panels at the right show the mixing (top) and synaptic weight matrix after training (bottom).

in Figure 3B, but it is not a sharp transition like the experimental result in Figure 3A. In addition, it does not transition at zero (ie: when $t_k - t_l = 0$), but at a time offset by the rise time of the EPSPs. In earlier experiments, in which we tranformed the gradient in (11) by an approximate inverse Hessian, to get an approximate Natural Gradient method, a sharp transition did emerge in simulations. However, the approximate inverse Hessian was singular, and we had to de-emphasise this result. It does suggest, however, that if the Natural Gradient transform can be usefully done on some variant of this learning rule, it may well be what accounts for the sharp transition effect of STDP.

## 3   Discussion

Although these derivations started out smoothly, the reader possibly shares the authors' frustration at the approximations involved here. Why isn't this simple, like ICA? Why don't we just have a nice *maximum spikelihood* model, ie: a density estimation algorithm for multivariate point processes, as ICA was a model in continuous space? We are going to be explicit about the problems now, and will propose a direction where the solution may lie. The over-riding problem is: we are unable to claim that in maximising $\log|\mathbf{T}|$, we are maximising the mutual information between inputs and outputs because:

**1. The Invertability Problem.** Algorithms such as ICA which maximise log Jacobians can only be called Infomax algorithms if the network transformation is both deterministic *and* invertible. The Spike Response Model is deterministic, but it is not invertible in general. When not invertible, the key formula (considering here vectors of input and output timings, $\mathbf{t}_{in}$ and $\mathbf{t}_{out}$)is transformed from simple to complex. ie:

$$p(\mathbf{t}_{out}) = \frac{p(\mathbf{t}_{in})}{|\mathbf{T}|} \text{ becomes } p(\mathbf{t}_{out}) = \int_{\text{solns } \mathbf{t}_{in}} \frac{p(\mathbf{t}_{in})}{|\mathbf{T}|} d\,\mathbf{t}_{in} \tag{12}$$

Thus when not invertible, we need to know the Jacobians of all the inputs that could have caused an output (called here 'solns'), something we simply don't know.

**2.  The 'Which Neuron' Problem.**  Instead of maximising the mutual information $I(\mathbf{t}_{out}, \mathbf{t}_{in})$, we should be maximising $I(\mathbf{ti}_{out}, \mathbf{ti}_{in})$, where the vector $\mathbf{ti}$ is the timing

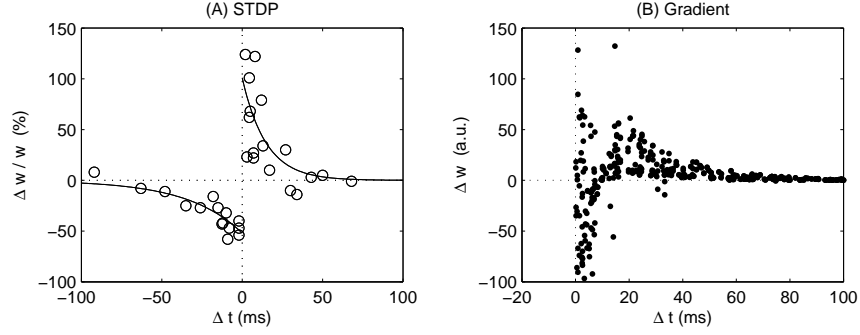

Figure 3: Dependence of synaptic modification on pre/post inter-spike interval. **Left (A):** From Froemke & Dan, Nature (2002)]. Dependence of synaptic modification on pre/post inter-spike interval in cat L2/3 visual cortical pyramidal cells in slice. Naturalistic spike trains. Each point represents one experiment. **Right (B):** According to Equation (11). Each point corresponds to an spike pair between approximately 100 input and 100 output spikes.

vector, $\mathbf{t}$, with the vector, $\mathbf{i}$, of corresponding neuron indices, concatenated. Thus, 'who spiked?' should be included in the analysis as it is part of the information.

**3. The Predictive Information Problem.** In ICA, since there was no time involved, we did not have to worry about mutual informations over time between inputs and outputs. But in the spiking model, output spikes may well have (predictive) mutual information with future input spikes, as well as the usual (causal) mutual information with past input spikes. The former has been entirely missing from our analysis so far.

These temporal and spatial infomation dependencies missing in our analysis so far, are thrown into a different light by a single empirical observation, which is that *Spike Timing-Dependent Plasticity is not just a feedforward computation like the Spike Response Model*. Specifically, there must be *at least a statistical, if not a causal*, relation between a real synapse's plasticity and its neuron's output spike timings, for Figure 3B to look like it does.

It seems we have to confront the need for both a 'memory' (or reconstruction) model, such as the $\mathbf{T}$ we have thus far dealt with, in which output spikes talk about past inputs, and a 'prediction' model, in which they talk about future inputs. This is most easily understood from the point of view of Barber & Agakov's variational Infomax algorithm [3]. They argue for optimising a lower bound on mutual information, which, for our neurons', would be expressed using an inverse model $\hat{p}$, as follows:

$$\widetilde{I}(\mathbf{ti}_{in}, \mathbf{ti}_{out}) = H(\mathbf{ti}_{in}) - \langle \log \hat{p}(\mathbf{ti}_{in}|\mathbf{ti}_{out}) \rangle_{p(\mathbf{ti}_{in},\mathbf{ti}_{out})} \leq I(\mathbf{ti}_{in}, \mathbf{ti}_{out}) \qquad (13)$$

In a feedforward model, $H(\mathbf{ti}_{in})$ may be disregarded in taking gradients, leading us to the optimisation of a *'memory-prediction' model* $\hat{p}(\mathbf{ti}_{in}|\mathbf{ti}_{out})$ related to something supposedly happening in dendrites, somas and at synapses. In trying to guess what this might be, it would be nice if the math worked out. We need a square Jacobian matrix, $\mathbf{T}$, so that $|\mathbf{T}| = \hat{p}(\mathbf{ti}_{in}|\mathbf{ti}_{out})$ can be our memory/prediction model. Now let's rename our feedforward timing Jacobian $\mathbf{T}$ ('up the dendritic trees'), as $\overrightarrow{\mathbf{T}}$, and let's fantasise that there is some, as yet unspecified, feedback Jacobian $\overleftarrow{\mathbf{T}}$ ('down the dendritic trees'), which covers electrotonic influences as they spread from soma to synapse, and which $\overrightarrow{\mathbf{T}}$ can be combined with by some operation '$\otimes$' to make things square. Imagine further, that doing this yields a memory/prediction model on the inputs. Then the $\mathbf{T}$ we are looking for is $\overrightarrow{\mathbf{T}} \otimes \overleftarrow{\mathbf{T}}$,

and the memory-prediction model is: $\hat{p}(\mathbf{ti}_{in}|\mathbf{ti}_{out}) = \left| \overrightarrow{\mathbf{T}} \otimes \overleftarrow{\mathbf{T}} \right|$

Ideally, the entries of $\overrightarrow{\mathbf{T}}$ should be as before, ie: $\overrightarrow{T}_{kl} = \partial t_k / \partial t_l$. What should the entries of $\overleftarrow{\mathbf{T}}$ be? Becoming just one step more concrete, suppose $\overleftarrow{\mathbf{T}}$ had entries $\overleftarrow{T}_{lk} = \partial c_l / \partial t_k$, where $c_l$ is some, as yet unspecified, value, or process, occuring at an input synapse when spike $l$ comes in. What seems clear is that $\otimes$ should combine the correctly tensorised forms of $\overrightarrow{\mathbf{T}}$ and $\overleftarrow{\mathbf{T}}$ (giving them each 4 indices $ijkl$), so that $\mathbf{T} = \overrightarrow{\mathbf{T}} \otimes \overleftarrow{\mathbf{T}}$ sums over the spikes $k$ and $l$ to give a $I \times J$ matrix, where $I$ is the number of output neurons, and $J$ the number of input neurons. Then our quantity, $\mathbf{T}$, would represent all dependencies of input neuronal activity on output activity, summed over spikes.

Further, we imagine that $\overleftarrow{\mathbf{T}}$ contains reverse (feedback) electrotonic transforms from soma to synapse $\overleftarrow{R}_{lk}$ that are somehow symmetrically related to the feedforward Spike Responses from synapse to soma, which we now rename $\overrightarrow{R}_{kl}$. Thinking for a moment in terms of somatic $k$ and synaptic $l$, voltages $V$, currents $I$ and linear cable theory, the synapse to soma transform, $\overrightarrow{R}_{kl}$ would be related to an *impedance* in $V_k = I_l \overrightarrow{Z}_{kl}$, while the soma to synapse transform, $\overleftarrow{R}_{lk}$ would be related to an *admittance* in $I_l = V_k \overleftarrow{Y}_{lk}$ [8]. The symmetry in these equations is that $\overrightarrow{Z}_{kl}$ is just the inverse conjugate of $\overleftarrow{Y}_{lk}$.

Finally, then, what is $c_l$? And what is its relation to the calcium concentration, $[\mathrm{Ca}^{2+}]_l$, at a synapse, when spike $l$ comes in? These questions naturally follow from considering the experimental data, since it is known that the calcium level at synapses is the critical integrating factor in determining whether potentiation or depression occurs [5].

## 4   Appendix: Gradient of $\log |\mathbf{T}|$ for the full Spike Response Model.

Here we give full details of the gradient for Gerstner's Spike Response Model [7]. This is a general model for which Integrate-and-Fire is a special case. In this model the effect of a presynaptic spike at time $t_l$ on the membrane potential at time $t$ is described by a post synaptic potential or spike response, which may also depend on the time that has passed since the last output spike $t_{k-1}$, hence the spike response is written as $R(t - t_{k-1}, t - t_l)$. This response is weighted by the synaptic strength $w_l$. Excitatory or inhibitory synapses are determined by the sign of $w_l$. Refractoriness is incorporated by adding a hyper-polarizing contribution (spike-afterpotential) to the membrane potential in response to the last preceding spike $\eta(t - t_{k-1})$. The membrane potential as a function of time is therefore given by

$$u(t) = \eta(t - t_{k-1}) + \sum_l w_l R(t - t_{k-1}, t - t_l). \tag{14}$$

We have ignored here potential contributions from external currents which can easily be included without modifying the following derivations. The output firing times $t_k$ are defined as the times for which $u(t)$ reaches firing threshold from below. We consider a dynamic threshold, $\vartheta(t - t_{k-1})$, which may depend on the time since that last spike $t_{k-1}$, together then output spike times are defined implicitly by:

$$t = t_k : u(t) = \vartheta(t - t_{k-1}) \text{ and } \frac{du(t)}{dt} > 0. \tag{15}$$

For this more general model $T_{kl}$ is given by

$$T_{kl} = \frac{dt_k}{dt_l} = -\left( \frac{\partial u}{\partial t_k} - \frac{\partial \vartheta}{\partial t_k} \right)^{-1} \frac{\partial u}{\partial t_l} = \frac{w_{kl} \dot{R}(t_k - t_{k-1}, t_k - t_l,)}{\dot{u}(t_k) - \dot{\vartheta}(t_k - t_{k-1})}, \tag{16}$$

where $\dot{R}(s, t), \dot{u}(t)$, and $\dot{\vartheta}(t)$ are derivatives with respect to $t$. The dependence of $T_{kl}$ on $t_{k-1}$ should be implicitly assumed. It has been omitted to simplify the notation.

Now we compute the derivative of $\log |\mathbf{T}|$ with respect to $w_{kl}$. For any matrix $\mathbf{T}$ we have $\partial \log |\mathbf{T}| / \partial T_{ab} = [\mathbf{T}^{-1}]_{ba}$. Therefore:

$$\frac{\partial \log |\mathbf{T}|}{\partial w_{kl}} = \sum_{ab} \frac{\partial \log |\mathbf{T}|}{\partial T_{ab}} \frac{\partial T_{ab}}{\partial w_{kl}} \sum_{ab} [\mathbf{T}^{-1}]_{ba} \frac{\partial T_{ab}}{\partial w_{kl}} . \tag{17}$$

Utilising the Kronecker delta $\delta_{ab} = (1 \text{ if } a = b, \text{ else } 0)$, the derivative of (16) with respect to $w_{kl}$ gives:

$$
\begin{aligned}
\frac{\partial T_{ab}}{\partial w_{kl}} &= \frac{\partial}{\partial w_{kl}} \left[ \frac{w_{ab}\dot{R}(t_a - t_{a-1}, t_a - t_b)}{\eta(t_a - t_{a-1}) + \sum_c w_{ac}\dot{R}(t_a - t_{a-1}, t_a - t_c) - \dot{\vartheta}(t_a - t_{a-1})} \right] \\
&= \delta_{ak}\delta_{bl} \frac{\dot{R}(t_a - t_{a-1}, t_a - t_b)}{\dot{u}(t_a) - \dot{\vartheta}(t_a - t_{a-1})} \\
&\quad - \frac{w_{ab}\dot{R}(t_a - t_{a-1}, t_a - t_b)\delta_{ak}\dot{R}(t_a - t_{a-1}, t_a - t_l)}{\left( \dot{u}(t_a) - \dot{\vartheta}(t_a - t_{a-1}) \right)^2} \\
&= \delta_{ak}T_{ab} \left[ \frac{\delta_{bl}}{w_{ab}} - \frac{T_{al}}{w_{al}} \right] .
\end{aligned} \tag{18}
$$

Therefore:

$$
\begin{aligned}
\frac{\partial \log |\mathbf{T}|}{\partial w_{kl}} &= \sum_{ab} [\mathbf{T}^{-1}]_{ba}\delta_{ak}T_{ab} \left[ \frac{\delta_{bl}}{w_{ab}} - \frac{T_{al}}{w_{al}} \right] \tag{19} \\
&= \frac{T_{kl}}{w_{kl}} \left( [\mathbf{T}^{-1}]_{lk} - \sum_b [\mathbf{T}^{-1}]_{bk}T_{kl} \right) = \frac{T_{kl}}{w_{kl}} \left( [\mathbf{T}^{-1}]_{lk} - 1 \right) . \tag{20}
\end{aligned}
$$

## Acknowledgments

We are grateful for inspirational discussions with Nihat Ay, Michael Eisele, Hong Hui Yu, Jim Crutchfield, Jeff Beck, Surya Ganguli, Sophiè Deneve, David Barber, Fabian Theis, Tony Zador and Arunava Banerjee. AJB thanks all RNI colleagues for many such discussions.

## References

[1] Amari S-I. 1997. Natural gradient works efficiently in learning, *Neural Computation*, 10, 251-276

[2] Banerjee A. 2001. On the Phase-Space Dynamics of Systems of Spiking Neurons. *Neural Computation*, 13, 161-225

[3] Barber D. & Agakov F. 2003. The IM Algorithm: A Variational Approach to Information Maximization. *Advances in Neural Information Processing Systems 16*, MIT Press.

[4] Bell A.J. & Sejnowski T.J. 1995. An information maximization approach to blind separation and blind deconvolution, *Neural Computation*, 7, 1129-1159

[5] Dan Y. & Poo M-m. 2004. Spike timing-dependent plasticity of neural circuits, *Neuron*, 44, 23-30

[6] Froemke R.C. & Dan Y. 2002. Spike-timing-dependent synaptic modification induced by natural spike trains. *Nature*, 28, 416: 433-8

[7] Gerstner W. & Kistner W.M. 2002. *Spiking neuron models*, Camb. Univ. Press

[8] Zador A.M., Agmon-Snir H. & Segev I. 1995. The morphoelectrotonic transform: a graphical approach to dendritic function, *J. Neurosci.*, 15(3): 1669-1682
